# An Information-Theoretic Framework for Understanding Saccadic Eye Movements

**Tai Sing Lee** *
Department of Computer Science
Carnegie Mellon University
Pittsburgh, PA 15213
*tai@cs.cmu.edu*

**Stella X. Yu**
Robotics Institute
Carnegie Mellon University
Pittsburgh, PA 15213
*stella@cnbc.cmu.edu*

## Abstract

In this paper, we propose that information maximization can provide a unified framework for understanding saccadic eye movements. In this framework, the mutual information among the cortical representations of the retinal image, the priors constructed from our long term visual experience, and a dynamic short-term internal representation constructed from recent saccades provides a map for guiding eye navigation. By directing the eyes to locations of maximum complexity in neuronal ensemble responses at each step, the automatic saccadic eye movement system greedily collects information about the external world, while modifying the neural representations in the process. This framework attempts to connect several psychological phenomena, such as pop-out and inhibition of return, to long term visual experience and short term working memory. It also provides an interesting perspective on contextual computation and formation of neural representation in the visual system.

## 1   Introduction

When we look at a painting or a visual scene, our eyes move around rapidly and constantly to look at different parts of the scene. Are there rules and principles that govern where the eyes are going to look next at each moment? In this paper, we sketch a theoretical framework based on information maximization to reason about the organization of saccadic eye movements.

Vision is fundamentally a Bayesian inference process. Given the measurement by the retinas, the brain's memory of eye positions and its prior knowledge of the world, our brain has to make an inference about what is where in the visual scene. The retina, unlike a camera, has a peculiar design. It has a small foveal region dedicated to high-resolution analysis and a large low-resolution peripheral region for monitoring the rest of the visual field. At about 2.5° visual angle away from the center of the fovea, visual acuity is already reduced by a half. When we 'look' (foveate) at a certain location in the visual scene, we direct our high-resolution fovea to analyze information in that location, taking a snap shot of the scene using our retina. Figure 1A-C illustrate what a retina would see at each fixation. It is immediately obvious that our retinal image is severely limited – it is clear only in the fovea and is very blurry in the surround, posing a severe constraint on the information available to our inference system. Yet, in our subjective experience, the world seems to be stable, coherent and complete in front of us. This is a paradox that have engaged philosophical and scientific debates for ages. To overcome the constraint of the retinal image, during perception, the brain actively moves the eyes around to (1) gather information to construct a mental image of the world, and (2) to make inference about the world based on this mental image. Understanding the forces that drive saccadic eye movements is important to elucidating the principles of active perception.

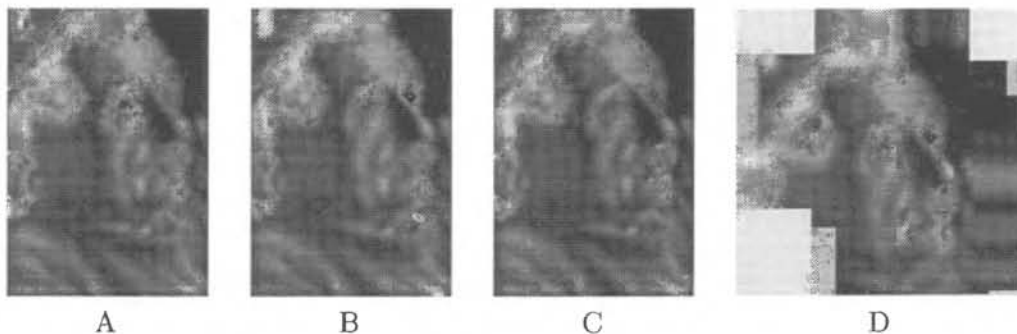

A          B          C          D

Figure 1. A-C: retinal images in three separate fixations. D: a mental mosaic created by integrating the retinal images from these three and other three fixations.

It is intuitive to think that eye movements are used to gather information. Eye movements have been suggested to provide a means for measuring the allocation of attention or the values of each kind of information in a particular context [16]. The basic assumption of our theory is that we move our eyes around to maximize our information intake from the world, for constructing the mental image and for making inference of the scene. Therefore, the system should always look for and attentively fixate at a location in the retinal image that is the most unusual or the most unexplained – and hence carries the maximum amount of information.

## 2   Perceptual Representation

How can the brain decide which part of the retinal image is more unusual? First of all, we know the responses of V1 simple cells, modeled well by the Gabor wavelet pyramid [3,7], can be used to reconstruct completely the retinal image. It is also well established that the receptive fields of these neurons developed in such a way as to provide a compact code for natural images [8,9,13,14]. The idea of compact code or sparse code, originally proposed by Barlow [2], is that early visual neurons capture the statistical correlations in natural scenes so that only a small number

of cells out of a large set will be activated to represent a particular scene at each moment. Extending this logic, we suggest that the complexity or the entropy of the neuronal ensemble response of a hypercolumn in V1 is therefore closely related to the strangeness of the image features being analyzed by the machinery in that hypercolumn. A frequent event will have a more compact representation in the neuronal ensemble response. Entropy is an information measure that captures the complexity or the variability of signals. The entropy of a neuronal ensemble in a hypercolumn can therefore be used to quantify the strangeness of a particular event.

A hypercolumn in the visual cortex contains roughly 200,000 neurons, dedicated to analyzing different aspects of the image in its 'visual window'. These cells are tuned to different spatial positions, orientations, spatial frequency, color disparity and other cues. There might also be a certain degree of redundancy, i.e. a number of neurons are tuned to the same feature. Thus a hypercolumn forms the fundamental computational unit for image analysis within a particular window in visual space. Each hypercolumn contains cells with receptive fields of different sizes, many significantly smaller than the aggregated 'visual window' of the hypercolumn. The entropy of a hypercolumn's ensemble response at a certain time $t$ is the sum of entropies of all the channels, given by,

$$H(u(R_{\vec{x}}, t)) = -\sum_{\theta, \sigma} \sum_{v} p(u(R_{\vec{x}}, v, \sigma, \theta, t)) \log_2 p(u(R_{\vec{x}}, v, \sigma, \theta, t))$$

where $u(R_{\vec{x}}, t)$ denotes the responses of all complex cell channels inside the visual window $R_{\vec{x}}$ of a hypercolumn at time $t$, computed within a 20 msec time window. $u(\vec{x}, \sigma, \theta, t)$ is the response of a V1 complex cell channel of a particular scale $\sigma$ and orientation $\sigma$ at spatial location $\vec{x}$ at $t$. $p(u(R_{\vec{x}}, v, \sigma, \theta, t))$ is the probability of cells in that channel within the visual window $R_{\vec{x}}$ of the hypercolumn firing $v$ number of spikes. $v$ can be computed as the power modulus of the corresponding simple cell channels, modeled by Gabor wavelets [see 7]. $\sum_{v} p(u(R_{\vec{x}}, v, \sigma, \theta, t)) = 1$. The probability $p(u(R_{\vec{x}}, v, \sigma, \theta, t))$ can be computed at each moment in time because of the variations in spatial position of the receptive fields of similar cell within the hypercolumn – hence the 'same' cells in the hypercolumn are analyzing different image patches, and also because of the redundancy of cells coding similar features.

The neurons' responses in a hypercolumn are subject to contextual modulation from other hypercolumns, partly in the form of lateral inhibition from cells with similar tunings. The net observed effect is that the later part of V1 neurons' response, starting at about 80 msec, exhibits differential suppression depending on the spatial extent and the nature of the surround stimulus. The more similar the surround stimulus is to the center stimuli, and the larger the spatial extent of the 'similar surround', the stronger is the suppressive effect [e.g. 6]. Simoncelli and Schwartz [15] have proposed that the steady state responses of the cells can be modeled by dividing the response of the cell (i.e. modeled by the wavelet coefficient or its power modulus) by a weighted combination of the responses of its spatial neighbors in order to remove the statistical dependencies between the responses of spatial neighbors. These weights are found by minimizing a predictive error between the center signal from the surround signals. In our context, this idea of predictive coding [see also 14] is captured by the concept of mutual information between the ensemble responses of the different hypercolumns as given below,

$$
\begin{aligned}
I(u(R_{\vec{x}}, t); u(\Omega_{\vec{x}}, t - \delta t_1)) &= H(u(R_{\vec{x}}, t)) - H(u(R_{\vec{x}}, t)|u(\Omega_{\vec{x}}, t - \delta t_1)) \\
&= \sum_{\sigma, \theta} \sum_{v_R, v_\Omega} [p(u(R_{\vec{x}}, v_R, \sigma, \theta, t), u(\Omega_{\vec{x}}, v_\Omega, \sigma, \theta, t)) \\
&\quad \log_2 \frac{p(u(R_{\vec{x}}, v_R, \sigma, \theta, t), u(\Omega_{\vec{x}}, v_\Omega, \sigma, \theta, t))}{p(u(R_{\vec{x}}, v_R, \sigma, \theta, t)), p(u(\Omega_{\vec{x}}, v_\Omega, \sigma, \theta, t))}].
\end{aligned}
$$

where $u(R_{\vec{x}}, t)$ is the ensemble response of the hypercolumn in question, and $u(\Omega_{\vec{x}}, t)$ is the ensemble response of the surrounding hypercolumns. $p(u(R_{\vec{x}}, v_R, \sigma, \theta, t))$ is the probability that cells of a channel in the center hypercolumn assumes the response value $v_R$ and $p(u(\Omega_{\vec{x}}, v_R, \sigma, \theta, t))$ the probability that cells of a similar channel in the surrounding hypercolumns assuming the response value $v_\Omega$. $t_1$ is the delay by which the surround information exerts its effect on the center hypercolumn. The mutual information $I$ can be computed from the joint probability of ensemble responses of the center and the surround.

The steady state responses of the V1 neurons, as a result of this contextual modulation, are said to be more correlated to perceptual pop-out than the neurons' initial responses [5,6]. The complexity of the steady state response in the early visual cortex is described by the following conditional entropy,

$$H(u(R_{\vec{x}}, t)|u(\Omega_{\vec{x}}, t - \delta t_1)) = H(u(R_{\vec{x}}, t)) - I(u(R_{\vec{x}}, t); u(\Omega_{\vec{x}}, t - \delta t_1)).$$

However, the computation in V1 is not limited to the creation of compact representation through surround inhibition. In fact, we have suggested that V1 plays an active role in scene interpretation particularly when such inference involves high resolution details [6]. Visual tasks such as the inference of contour and surface likely involve V1 heavily. These computations could further modify the steady state responses of V1, and hence the control of saccadic eye movements.

## 3 Mental Mosaic Representation

The perceptual representation provides the basic force for the brain to steer the high resolution fovea to locations of maximum uncertainty or maximum signal complexity. Foveation captures the maximum amount of available information in a location. Once a location is examined by the fovea, its information uncertainty is greatly reduced. The eyes should move on and not to return to the same spot within a certain period of time. This is called the 'inhibition of return'.

How can we model this reduction of interest? We propose that the mind creates a mental mosaic of the scene in order to keep track of the information that have been gathered. By mosaic, we mean that the brain can assemble successive retinal images obtained from multiple fixations into a coherent mental picture of the scene. Figure 1D provides an example of a mental mosaic created by combining information from the retinal images from 6 fixations. Whether the brain actually keeps such a mental mosaic of the scene is currently under debate. McConkie and Rayner [10] had suggested the idea of an integrative visual buffer to integrate information across multiple saccades. However, numerous experiments demonstrated we actually remember relatively little across saccades [4]. This lead to the idea that brain may not need an explicit internal representation of the world. Since the world is always out there, the brain can access whatever information it needs at the appropriate details by moving the eyes to the appropriate place at the appropriate time. The subjective feeling of a coherent and a complete world in front of us is a mere illusion [e.g. 1].

The mental mosaic represented in Figure 1D might resemble McConkie and Rayner's theory superficially. But the existence of such a detailed high-resolution buffer with a large spatial support in the brain is rather biologically implausible. Rather, we think that the information corresponding to the mental mosaic is stored in an *interpreted* and *semantic* form in a mesh of Bayesian belief networks in the brain (e.g. involving PO, IT and area 46). This distributed semantic representation of

the mental mosaic, however, is capable of generating detailed (sometimes false) imagery in early visual cortex using the massive recurrent convergent feedback from the higher areas to V1. However, because of the limited support provided by V1 machinery, the instantiation of mental imagery in V1 has to be done sequentially one 'retinal image' frame at a time, presumably in conjunction with eye movement, even when the eyes are closed. This might explain why vivid visual dream is always accompanied by rapid eye movement in REM sleep. The mental mosaic accumulates information from the retinal images up to the last fixation and can provide prediction on what the retina will see in the current fixation. For each $u(\vec{x}, \sigma, \theta)$ cell, there is a corresponding effective prediction signal $m(\vec{x}, \sigma, \theta)$ fed back from the mental mosaic.

This prediction signal can reduce the conditional entropy or complexity of the ensemble response in the perceptual representation by discounting the mutual information between the ensemble response to the retinal image and the mental mosaic prediction as follow,

$$H(u(R_{\vec{x}}, t)|m(R_{\vec{x}}, t - \delta t_2)) = H(u(R_{\vec{x}}, t)) - I(u(R_{\vec{x}}, t), m(R_{\vec{x}}, t - \delta t_2))$$

where $\delta t_2$ is the transmission delay from the mental mosaic back to V1.

At places where the fovea has visited, the mental mosaic representation has high resolution information and $m(\vec{x}, \sigma, \theta, t - \delta t_2)$ can explain $u(\vec{x}, \sigma, \theta, t)$ fully. Hence, the mutual information is high at those hypercolumns and the conditional entropy $H(u(R_{\vec{x}}, t)|m(R_{\vec{x}}, t - \delta t_2))$ is low, with two consequences: (1) the system will not get the eyes stuck at a particular location; once the information at $\vec{x}$ is updated to the mental mosaic, the system will lose interest and move on; (2) the system will exhibit 'inhibition of return' as the information in the visited locations are fully predicted by the mental mosaic. Also, from this standpoint, the 'habituation dynamics' often observed in visual neurons when the same stimulus is presented multiple times might not be simply due to neuro-chemical fatigue, but might be understood in terms of mental mosaic being updated and then fed back to explain the perceptual representation in V1. The mental mosaic is in effect our short-term memory of the scene. It has a forgetting dynamics, and needs to be periodically updated. Otherwise, it will rapidly fade away.

## 4   Overall Reactive Saccadic Behaviors

Now, we can combine the influence of the two predictive processes to arrive at a discounted complexity measure of the hypercolumn's ensemble response:

$$\begin{aligned} H(u(R_{\vec{x}}, t)|u(\Omega_{\vec{x}}, t - \delta t_1), m(R_{\vec{x}}, t - \delta t_2)) = & H(u(R_{\vec{x}}, t)) \\ & - I(u(R_{\vec{x}}, t); u(\Omega_{\vec{x}}, t - \delta t_1)) \\ & - I(u(R_{\vec{x}}, t); m(R_{\vec{x}}, t - \delta t_2)) \\ & + I(u(\Omega_{\vec{x}}, t - \delta t_1); m(R_{\vec{x}}, t - \delta t_2)) \end{aligned}$$

If we can assume the long range surround priors and the mental mosaic short term memory are independent processes, we can leave out the last term, $I(u(\Omega_{\vec{x}}, t - \delta t_1); m(R_{\vec{x}}, t - \delta t_2))$, of the equation.

The system, after each saccade, will evaluate the new retinal scene and select the location where the perceptual representation has the maximum conditional entropy. To maximize the information gain, the system must constantly search for and make a saccade to the locations of maximum uncertainty (or complexity) computed from

the hypercolumn ensemble responses in V1 at each fixation. Unless the number of saccades is severely limited, this locally greedy algorithm, coupled the inhibition of return mechanism, will likely steer the system to a relatively optimal global sampling of the world – in the sense that the average information gain per saccade is maximized, and the mental mosaic's dissonance with the world is minimized.

## 5    Task-dependent schema Representation

However, human eye movements are not simply controlled by the generic information in a bottom-up fashion. Yarbus [16] has shown that, when staring at a face, subjects' eyes tend to go back to the same locations (eyes, mouth) over and over again. Further, he showed that when asked different questions, subjects exhibited different kinds of scan-paths when looking at the same picture. Norton and Stark [12] also showed that eye movements are not random, but often exhibit repetitive or even idiosyncratic path patterns.

To capture these ideas, we propose a third representation, called task schema, to provide the necessary *top-down* information to bias the eye movement control. It specifies the learned or habitual scan-paths for a particular task in a particular context or assigns weights to different types of information. Given that we arenot mostly unconscious of the scan-path patterns we are making, these task-sensitive or context-sensitive habitual scan-patterns might be encoded at the levels of motor programs, and be downloaded when needed without our conscious control. These motor programs for scan-paths can be trained from reinforcement learning. For example, since the eyes and the mouths convey most of the emotional content of a facial expression, a successful interpretation of another person's emotion could provide the reward signal to reinforce the motor programs just executed or the fixations to certain facial features. These unconscious scan-path motor programs could provide the additional modulation to automatic saccadic eye movement generation.

## 6    Discussion

In this paper, we propose that information maximization might provide a theoretical framework to understand the automatic saccadic eye movement behaviors in human. In this proposal, each hypercolumn in V1 is considered a fundamental computational unit. The relative complexity or entropy of the neuronal ensemble response in the V1 hypercolumns, discounted by the predictive effect of the surround, higher order representations and working memory, creates a force field to guide eye navigation.

The framework we sketched here bridge natural scene statistics to eye movement control via the more established ideas of sparse coding and predictive coding in neural representation. Information maximization has been suggested to be a possible explanation for shaping the receptive fields in the early visual cortex according to the statistics of natural images [8,9,13,14] to create a minimum-entropy code [2,3]. As a result, a frequent event is represented efficiently with the response of a few neurons in a large set, resulting in a lower hypercolumn ensemble entropy, while unusual events provoke ensemble responses of higher complexity. We suggest that higher complexity in ensemble responses will arouse attention and draw scrutiny by the eyes, forcing the neural representation to continue adapting to the statistics of the natural scenes. The formulation here also suggests that information maximization might provide an explanation for the formation of horizontal predictive network in V1 as well as higher order internal representations, consistent with the ideas of predictive coding [11, 14, 15]. Our theory hence predicts that the adaptation of the

neural representations to the statistics of natural scenes will lead to the adaptation of·saccadic eye movement behaviors.

## Acknowledgements

The authors have been supported by a grant from the McDonnell Foundation and a NSF grant (LIS 9720350). Yu is also being supported in part by a grant to Takeo Kanade.

## Footnotes

*Both authors are members of the Center for the Neural Basis of Cognition – a joint center between University of Pittsburgh and Carnegie Mellon University. Address: Rm 115, Mellon Institute, Carnegie Mellon University, Pittsburgh, PA 15213.

## References

[1] Ballard, D. Hayhoe, M.M. Pook, P.K. & Rao, R.P.N. (1997). Deictic codes for the embodiment of cognition. *Behavioral and Brain Science*, **20:4**, December, 723-767.

[2] Barlow, H.B. (1989). Unsupervised learning. *Neural Computation*, **1**, 295-311.

[3] Daugman, J.G. (1989). Entropy reduction and decorrelation in visual coding by oriented neural receptive fields. *IEEE Transactions on Biomedical Engineering 36:*, 107-114.

[4] Irwin, D. E, 1991. Information Integration across Saccadic Eye Movements. *Cognitive Psychology*, 23(3):420-56.

[5] Knierim, J. & Van Essen, D.C. Neural response to static texture patterns in area V1 of macaque monkey. *J. Neurophysiology*, **67:** 961-980.

[6] Lee, T.S., Mumford, D., Romero R. & Lamme, V.A.F. (1998). The role of primary visual cortex in higher level vision. *Vision Research* **38**, 2429-2454.

[7] Lee, T.S. (1996). Image representation using 2D Gabor wavelets. *IEEE Transaction of Pattern Analysis and Machine Intelligence.* **18:10**, 959-971.

[8] Lewicki, M. & Olshausen, B. (1998). Inferring sparse, overcomplete image codes using an efficient coding framework. In *Advances in Neural Information Processing System 10*, M. Jordan, M. Kearns and S. Solla (eds). MIT Press.

[9] Linsker, R. (1989). How to generate ordered maps by maximizing the mutual information between input and output signals. *Neural Computation*, **1:** 402-411.o

[10] McConkie, G.W. & Rayner, K. (1976). Identifying the span of effective stimulus in reading. Literature review and theories of reading. In H. Singer and R.B. Ruddell (Eds), *Theoretical models and processes of reading*, 137-162. Newark, D.E.: International Reading Association.

[11] Mumford, D. (1992). On the computational architecture of the neocortex II. *Biological cybernetics*, **66**, 241-251.

[12] Norton, D. and Stark, L. (1971) Eye movements and visual perception. *Scientific American*, **224**, 34-43.

[13] Olshausen, B.A., & Field, D.J. (1996), Emergence of simple cell receptive field properties by learning a sparse code for natural images. *Nature*, **381:** 607-609.

[14] Rao R., & Ballard, D.H. (1999). Predictive coding in the visual cortex: a functional interpretation of some extra-classical receptive field effects. *Nature Neuroscience*, **2:1** 79-87.

[15] Simoncelli, E.P. & Schwartz, O. (1999). Modeling surround suppression in V1 neurons with a statistically-derived normalization model. In *Advances in Neural Information Processing Systems 11*, . M.S. Kearns, S.A. Solla, and D.A. Cohn (eds). MIT Press.

[16] Yarbus, A.L. (1967). Eye movements and vision. Plenum Press.